# Distributed Inference in Dynamical Systems

**Stanislav Funiak    Carlos Guestrin**
Carnegie Mellon University

**Mark Paskin**
Google

**Rahul Sukthankar**
Intel Research

## Abstract

We present a robust distributed algorithm for approximate probabilistic inference in dynamical systems, such as sensor networks and teams of mobile robots. Using assumed density filtering, the network nodes maintain a tractable representation of the belief state in a distributed fashion. At each time step, the nodes coordinate to condition this distribution on the observations made throughout the network, and to advance this estimate to the next time step. In addition, we identify a significant challenge for probabilistic inference in dynamical systems: message losses or network partitions can cause nodes to have inconsistent beliefs about the current state of the system. We address this problem by developing distributed algorithms that guarantee that nodes will reach an informative consistent distribution when communication is re-established. We present a suite of experimental results on real-world sensor data for two real sensor network deployments: one with 25 cameras and another with 54 temperature sensors.

## 1   Introduction

Large-scale networks of sensing devices have become increasingly pervasive, with applications ranging from sensor networks and mobile robot teams to emergency response systems. Often, nodes in these networks need to perform **probabilistic dynamic inference** to combine a sequence of local, noisy observations into a global, joint estimate of the system state. For example, robots in a team may combine local laser range scans, collected over time, to obtain a global map of the environment; nodes in a camera network may combine a set of image sequences to recognize moving objects in a heavily cluttered scene. A simple approach to probabilistic dynamic inference is to collect the data to a central location, where the processing is performed. Yet, collecting all the observations is often impractical in large networks, especially if the nodes have a limited supply of energy and communicate over a wireless network. Instead, the nodes need to collaborate, to solve the inference task in a distributed manner. Such distributed inference techniques are also necessary in online control applications, where nodes of the network need estimates of the state in order to make decisions.

Probabilistic dynamic inference can often be efficiently solved when all the processing is performed centrally. For example, in linear systems with Gaussian noise, the inference tasks can be solved in a closed form with a Kalman Filter [3]; for large systems, **assumed density filtering** can often be used to approximate the filtered estimate with a tractable distribution (*c.f.*, [2]). Unfortunately, distributed dynamic inference is substantially more challenging. Since the observations are distributed across the network, nodes must coordinate to incorporate each others' observations and propagate their estimates from one time step to the next. Online operation requires the algorithm to degrade gracefully when nodes run out of processing time before the observations propagate throughout the network. Furthermore, the algorithm needs to robustly address node failures and interference that may partition the communication network into several disconnected components.

We present an efficient distributed algorithm for dynamic inference that works on a large family of processes modeled by dynamic Bayesian networks. In our algorithm, each node maintains a (possibly approximate) marginal distribution over a subset of state variables, conditioned on the measurements made by the nodes in the network. At each time step, the nodes condition on the observations, using a modification of the robust (static) distributed inference algorithm [7], and then advance their estimates to the next time step locally. The algorithm guarantees that, with sufficient communication at each time step, the nodes obtain the same solution as the corresponding centralized algorithm [2]. Before convergence, the algorithm introduces principled approximations in the form of independence assertions in the node estimates and in the transition model.

In the presence of unreliable communication or high latency, the nodes may not be able to condition their estimates on all the observations in the network, e.g., when interference causes a network partition, or when high latency prevents messages from reaching every node. Once the estimates are advanced to the next time step, it is difficult to condition on the observations made in the past [10]. Hence, the beliefs at the nodes may be conditioned on different evidence and no longer form a consistent global probability distribution over the state space. We show that such inconsistencies can lead to poor results when nodes attempt to combine their estimates. Nevertheless, it is often possible to use the inconsistent estimates to form an informative globally consistent distribution; we refer to this task as **alignment**. We propose an online algorithm, **optimized conditional alignment** (OCA), that obtains the global distribution as a product of conditionals from local estimates and optimizes over different orderings to select a global distribution of minimal entropy. We also propose an alternative, more global optimization approach that minimizes a KL divergence-based criterion and provides accurate solutions even when the communication network is highly fragmented.

We present experimental results on real-world sensor data, covering sensor calibration [7] and distributed camera localization [5]. These results demonstrate the convergence properties of the algorithm, its robustness to message loss and network partitions, and the effectiveness of our method at recovering from inconsistencies.

Distributed dynamic inference has received some attention in the literature. For example, particle filtering (PF) techniques have been applied to these settings: Zhao et al. [11] use (mostly) independent PFs to track moving objects, and Rosencrantz et al. [10] run PFs in parallel, sharing measurements as appropriate. Pfeffer and Tai [9] use loopy belief propagation to approximate the estimation step in a continuous-time Bayesian network. When compared to these techniques, our approach addresses several additional challenges: we do not assume point-to-point communication between nodes, we provide robustness guarantees to node failures and network partitions, and we identify and address the belief inconsistency problem that arises in distributed systems.

## 2 The distributed dynamic inference problem

Following [7], we assume a network model where each node can perform local computations and communicate with other nodes over some channel. The nodes of the network may change over time: existing nodes can fail, and new nodes may be introduced. We assume a message-level error model: messages are either received without error, or they are not received at all. The likelihood of successful transmissions (link qualities) are unknown and can change over time, and link qualities of several node pairs may be correlated.

We model the system as a dynamic Bayesian network (DBN). A DBN consists of a set of **state processes**, $\mathbf{X} = \{X_1, \ldots, X_L\}$ and a set of observed **measurement processes** $\mathbf{Z} = \{Z_1, \ldots, Z_K\}$; each measurement process $Z_k$ corresponds to one of the sensors on one of the nodes. State processes are not associated with unique nodes. A DBN defines a joint probability model over steps $1 \ldots T$ as

$$p(\mathbf{X}^{(1:T)}, \mathbf{Z}^{(1:T)}) = \underbrace{p(\mathbf{X}^{(1)})}_{\text{initial prior}} \times \prod_{t=2}^{T} \underbrace{p(\mathbf{X}^{(t)} \mid \mathbf{X}^{(t-1)})}_{\text{transition model}} \times \prod_{t=1}^{T} \underbrace{p(\mathbf{Z}^{(t)} \mid \mathbf{X}^{(t)})}_{\text{measurement model}}.$$

The initial prior is given by a factorized probability model $p(\mathbf{X}^{(1)}) \propto \prod_h \psi(\mathbf{A}_h^{(1)})$, where each $\mathbf{A}_h \subseteq \mathbf{X}$ is a subset of the state processes. The transition model factors as $\prod_{i=1}^{L} p(X_i^{(t)} \mid \mathbf{Pa}[X_i^{(t)}])$, where $\mathbf{Pa}[X_i^{(t)}]$ are the parents of $X_i^{(t)}$ in the previous time step. The measurement model factors as $\prod_{k=1}^{K} p(Z_k^{(t)} \mid \mathbf{Pa}[Z_k^{(t)}])$, where $\mathbf{Pa}[Z_k^{(t)}] \subseteq \mathbf{X}^{(t)}$ are the parents of $Z_k^{(t)}$ in the current time step.

In the distributed dynamic inference problem, each node $n$ is associated with a set of processes $\mathbf{Q}_n \subseteq \mathbf{X}$; these are the processes about which node $n$ wishes to reason. The nodes need to collaborate so that each node can obtain (an approximation to) the posterior distribution over $\mathbf{Q}_n^{(t)}$ given all measurements made in the network up to the current time step $t$: $p(\mathbf{Q}_i^{(t)} \mid \mathbf{z}^{(1:t)})$. We assume that node clocks are synchronized, so that transitions to the next time step are simultaneous.

## 3 Filtering in dynamical systems

The goal of (centralized) **filtering** is to compute the **posterior distribution** $p(\mathbf{X}^{(t)} \mid \mathbf{z}^{(1:t)})$ for $t = 1, 2, \ldots$ as the observations $\mathbf{z}^{(1)}, \mathbf{z}^{(2)}, \ldots$ arrive. The basic approach is to recursively compute $p(\mathbf{X}^{(t+1)} \mid \mathbf{z}^{(1:t)})$ from $p(\mathbf{X}^{(t)} \mid \mathbf{z}^{(1:t-1)})$ in three steps:

1. **Estimation**: $p(\mathbf{X}^{(t)} \mid \mathbf{z}^{(1:t)}) \propto p(\mathbf{X}^{(t)} \mid \mathbf{z}^{(1:t-1)}) \times p(\mathbf{z}^{(t)} \mid \mathbf{X}^{(t)})$;
2. **Prediction**: $p(\mathbf{X}^{(t)}, \mathbf{X}^{(t+1)} \mid \mathbf{z}^{(1:t)}) = p(\mathbf{X}^{(t)} \mid \mathbf{z}^{(1:t)}) \times p(\mathbf{X}^{(t+1)} \mid \mathbf{X}^{(t)})$;
3. **Roll-up**: $p(\mathbf{X}^{(t+1)} \mid \mathbf{z}^{(1:t)}) = \int p(\mathbf{x}^{(t)}, \mathbf{X}^{(t+1)} \mid \mathbf{z}^{(1:t)}) \, d\mathbf{x}^{(t)}$.

Exact filtering in DBNs is usually expensive or intractable because the belief state rapidly loses all conditional independence structure. An effective approach, proposed by Boyen and Koller [2], hereby denoted "B&K98", is to periodically project the exact posterior to a distribution that satisfies independence assertions encoded in a junction tree [3]. Given a junction tree $T$, with cliques $\{\mathbf{C}_i\}$ and separators $\{\mathbf{S}_{i,j}\}$, the projection operation amounts to computing the clique marginals, hence the filtered distribution is approximated as

$$p(\mathbf{X}^{(t)} \,|\, \mathbf{z}^{(1:t-1)}) \approx \tilde{p}(\mathbf{X}^{(t)} \,|\, \mathbf{z}^{(1:t-1)}) = \frac{\prod_{i \in N_T} \tilde{p}(\mathbf{C}_i^{(t)} \,|\, \mathbf{z}^{(1:t-1)})}{\prod_{\{i,j\} \in E_T} \tilde{p}(\mathbf{S}_{i,j}^{(t)} \,|\, \mathbf{z}^{(1:t-1)})}, \tag{1}$$

where $N_T$ and $E_T$ are the nodes and edges of $T$, respectively. With this representation, the estimation step is implemented by multiplying each observation likelihood $p(z_k^{(t)} \,|\, \mathbf{Pa}[Z_k^{(t)}])$ to a clique marginal; the clique and separator potentials are then recomputed with message passing, so that the posterior distribution is once again written as a ratio of clique and separator marginals: $\tilde{p}(\mathbf{X}^{(t)} \,|\, \mathbf{z}^{(1:t)}) = \left[ \prod_{i \in N_T} \tilde{p}(\mathbf{C}_i^{(t)} \,|\, \mathbf{z}^{(1:t)}) \right] \Big/ \left[ \prod_{\{i,j\} \in E_T} \tilde{p}(\mathbf{S}_{i,j}^{(t)} \,|\, \mathbf{z}^{(1:t)}) \right]$. The prediction step is performed independently for each clique $\mathbf{C}_i^{(t+1)}$: we multiply $\tilde{p}(\mathbf{X}^{(t)} \,|\, \mathbf{z}^{(1:t)})$ with the transition model $p(X^{(t+1)} \,|\, \mathbf{Pa}[X^{(t+1)}])$ for each variable $X^{(t+1)} \in \mathbf{C}_i^{(t+1)}$ and, using variable elimination, compute the marginals over the clique at the next time step $p(\mathbf{C}_i^{(t+1)} \,|\, \mathbf{z}^{(1:t)})$.

## 4 Approximate distributed filtering

In principle, the centralized filtering approach described in the previous section could be applied to a distributed system, e.g., by communicating the observations made in the network to a central location that performs all computations, and distributing the answer to every node in the network. While conceptually simple, this approach has substantial drawbacks, including the high communication bandwidth, the introduction of a single point of failure to the system, and the fact that nodes do not have valid estimates when the network is partitioned. In this section, we present a distributed filtering algorithm where each node obtains an approximation to the posterior distribution over subset of the state variables. Our estimation step builds on the robust distributed inference algorithm of Paskin *et al.* [7, 8], while the prediction, roll-up, and projection steps are performed locally at each node.

### 4.1 Estimation as a robust distributed probabilistic inference

In the distributed inference approach of Paskin *et al.* [8], the nodes collaborate so that each node $n$ can obtain the posterior distribution over some set of variables $\mathbf{Q}_n$ given all measurements made throughout the network. In our setting, $\mathbf{Q}_n$ contains the variables in a subset $L_n$ of the cliques used in our assumed density representation. In their architecture, nodes form a distributed data structure along a routing tree in the network, where each node in this tree is associated with a cluster of variables $\mathbf{D}_n$ that includes $\mathbf{Q}_n$, as well as any other variables, needed to preserve the flow of information between the nodes, a property equivalent to the **running intersection property** in junction trees [3]. We refer to this tree as the **network junction tree**, and, for clarity, we refer to the junction tree used for the assumed density as the **external junction tree**.

Using this architecture, Paskin and Guestrin developed a **robust distributed probabilistic inference** algorithm, RDPI [7], for static inference settings, where nodes compute the posterior distribution $p(\mathbf{Q}_n \,|\, \mathbf{z})$ over $\mathbf{Q}_n$ given all measurements throughout the network $\mathbf{z}$. RDPI provides two crucial properties: **convergence**, if there are no network partitions, these distributed estimates converge to the true posteriors; and, **smooth degradation** even before convergence, the estimates provide a principled approximation to the true posterior (which introduces additional independence assertions).

In RDPI, each node $n$ maintains the current belief $\beta_n$ of $p(\mathbf{Q}_n \,|\, \mathbf{z})$. Initially, node $n$ knows only the marginals of the prior distribution $\{p(\mathbf{C}_i) : i \in L_n\}$ for a subset of cliques $L_n$ in the external junction tree, and its local observation model $p(z_n \,|\, \mathbf{Pa}[Z_n])$ for each of its sensors. We assume that $\mathbf{Pa}[Z_n] \subseteq \mathbf{C}_i$ for some $i \in L_n$; thus, $\beta_n$ is represented as a collection of priors over cliques of variables, and of observation likelihood functions over these variables. Messages are then sent between neighboring nodes, in an analogous fashion to the sum-product algorithm for junction trees [3]. However, messages in RDPI are always represented as a collection of priors $\{\pi_i(\mathbf{C}_i)\}$ over cliques of variables $\mathbf{C}_i$, and of measurement likelihood functions $\{\lambda_i(\mathbf{C}_i)\}$ over these cliques. This decomposition into prior and likelihood factors is the key to the robustness properties of the algorithm [7]. With sufficient communication, $\beta_n$ converges to $p(\mathbf{Q}_n \,|\, \mathbf{z})$.

In our setting, at each time step $t$, each prior $\pi_i(\mathbf{C}_i^{(t)})$ is initialized to $p(\mathbf{C}_i^{(t)} \,|\, \mathbf{z}^{(1:t-1)})$. The likelihood functions are similarly initialized to $\lambda_i(\mathbf{C}_i^{(t)}) = p(z_i^{(t)} \,|\, \mathbf{C}_i^{(t)})$, if some sensor makes an

observation about these variables, or to 1 otherwise. Through message passing $\beta_n$ converges to $\tilde{p}(\mathbf{Q}_n^{(t)} \,|\, \mathbf{z}^{(1:t)})$. An important property of RDPI that will be useful in the remainder of the paper is:

**Property 1.** *Let $\beta_n$ be the result computed by the* RDPI *algorithm at convergence at node $n$. Then the cliques in $\beta_n$ form a subtree of an external junction tree that covers $\mathbf{Q}_n$.*

## 4.2 Prediction, roll-up and projection

The previous section shows that the estimation step can be implemented in a distributed manner, using RDPI. At convergence, each node $n$ obtains the calibrated marginals $\tilde{p}(\mathbf{C}_i^{(t)} \,|\, \mathbf{z}^{(1:t)})$, for $i \in L_n$. In order to advance to the next time step, each node must perform prediction and roll-up, obtaining the marginals $\tilde{p}(\mathbf{C}_i^{(t+1)} \,|\, \mathbf{z}^{(1:t)})$. Recall from Section 3 that, in order to compute a marginal $\tilde{p}(\mathbf{C}_i^{(t+1)} \,|\, \mathbf{z}^{(1:t)})$, this node needs $\tilde{p}(\mathbf{X}^{(t)} \,|\, \mathbf{z}^{(1:t)})$. Due to the conditional independencies encoded in $\tilde{p}(\mathbf{X}^{(t)} \,|\, \mathbf{z}^{(1:t)})$, it is sufficient to obtain a subtree of the external junction tree that covers the parents $\mathbf{Pa}[\mathbf{C}_i^{(t+1)}]$ of all variables in the clique. The next time step marginal $\tilde{p}(\mathbf{C}_i^{(t+1)} \,|\, \mathbf{z}^{(1:t)})$ can then be computed by multiplying this subtree with the transition model $p(X^{(t+1)} \,|\, \mathbf{Pa}[X^{(t+1)}])$ for each $X^{(t+1)} \in \mathbf{C}_i^{(t+1)}$ and eliminating all variables but $\mathbf{C}_i^{(t+1)}$ (recall that $\mathbf{Pa}[X^{(t+1)}] \subseteq \mathbf{X}^{(t)}$).

This procedure suggests the following distributed implementation of prediction, roll-up, and projection: after completing the estimation step, each node selects a subtree of the (global) external junction tree that covers $\mathbf{Pa}[\mathbf{C}_i^{(t+1)}]$ and collects the marginals of this tree from other nodes in the network. Unfortunately, it is unclear how to allocate the running time between estimation and collection of marginals in time-critical applications, when the estimation step may not run to completion. Instead, we propose a simple approach that performs both steps at once: run the distributed inference algorithm, described in the previous section, to obtain the posterior distribution over the *parents* of each clique maintained at the node. This task can be accomplished by ensuring that these parent variables are included in the query variables of node $n$: $\mathbf{Pa}[\mathbf{C}_i^{(t+1)}] \subseteq \mathbf{Q}_n, \forall i \in L_n$.

When the estimation step cannot be run to convergence within the allotted time, the variables $\mathrm{Scope}[\beta_n]$ covered by the distribution $\beta_n$ that node $n$ obtains may not cover the entire parent set $\mathbf{Pa}[\mathbf{C}_i^{(t+1)}]$. In this case, multiplying in the standard transition model is equivalent to assuming an uniform prior for the missing variables, which can lead to very poor solutions in practice. When the transition model is learned from data, $p(X^{(t+1)} \,|\, \mathbf{Pa}[X^{(t+1)}])$ is usually computed from the empirical distribution $\hat{p}(X^{(t+1)}, \mathbf{Pa}[X^{(t+1)}])$, e.g., $p^{MLE}(X^{(t+1)} \,|\, \mathbf{Pa}[X^{(t+1)}]) = \hat{p}(X^{(t+1)}, \mathbf{Pa}[X^{(t+1)}])/\hat{p}(\mathbf{Pa}[X^{(t+1)}])$. Building on these empirical distributions, we can obtain an improved solution for the prediction and roll-up steps, when we do not have a distribution over the entire parent set $\mathbf{Pa}[\mathbf{C}_i^{(t+1)}]$. Specifically, we obtain a valid approximate transition model $\tilde{p}(X^{(t+1)} \,|\, \mathbf{W}^{(t)})$, where $\mathbf{W}^{(t)} = \mathrm{Scope}[\beta_n] \cap \mathbf{Pa}[X^{(t+1)}]$, online by simply marginalizing the empirical distribution $\hat{p}(X^{(t+1)}, \mathbf{Pa}[X^{(t+1)}])$ down to $\hat{p}(X^{(t+1)}, \mathbf{W}^{(t)})$. This procedure is equivalent to introducing an additional independence assertion to the model: at time step $t + 1$, $X^{(t+1)}$ is independent of $\mathbf{Pa}[X^{(t+1)}] - \mathbf{W}^{(t)}$, given $\mathbf{W}^{(t)}$.

## 4.3 Summary of the algorithm

Our distributed approximate filtering algorithm can be summarized as follows:

- Using the architecture in [8], construct a network junction tree s.t. the query variables $\mathbf{Q}_n$ at each node $n$ cover $\left( \bigcup_{i \in L_n} \mathbf{C}_i^{(t)} \right) \cup \left( \bigcup_{i \in L_n} \mathbf{Pa}[\mathbf{C}_i^{(t+1)}] \right)$.
- For $t = 1, 2, \ldots$, at each node $n$,
  - run RDPI [7] until the end of step $t$, obtaining a (possibly approximate) belief $\beta_n$;
  - for each $X^{(t+1)} \in \mathbf{C}_i^{(t+1)}$, $i \in L_n$, compute an approximate transition model $\tilde{p}(X^{(t+1)} \,|\, \mathbf{W}_X^{(t)})$, where $\mathbf{W}_X^{(t)} = \mathrm{Scope}[\beta_n] \cap \mathbf{Pa}[X^{(t+1)}]$;
  - for each clique $\mathbf{C}_i^{(t+1)}$, $i \in L_n$, compute the clique marginal $\tilde{p}(\mathbf{C}_i^{(t+1)} \,|\, \mathbf{z}^{(1:t)})$ from $\beta_n$ and from each $\tilde{p}(X^{(t+1)} \,|\, \mathbf{W}_X^{(t)})$, locally, using variable elimination.

Using the convergence properties of the RDPI algorithm, we prove that, given sufficient communication, our distributed algorithm obtains the same solution as the centralized B&K98 algorithm:

**Theorem 1.** *For a set of nodes running our distributed filtering algorithm, if at each time step there is sufficient communication for the* RDPI *algorithm to converge, and the network is not partitioned, then, for each node $n$, for each clique $i \in L_n$, the distribution $\tilde{p}(\mathbf{C}_i^{(t)} \,|\, \mathbf{z}^{(1:t-1)})$ obtained by node $n$ is equal to the distribution obtained by the* B&K98 *algorithm with assumed density given by $T$.*

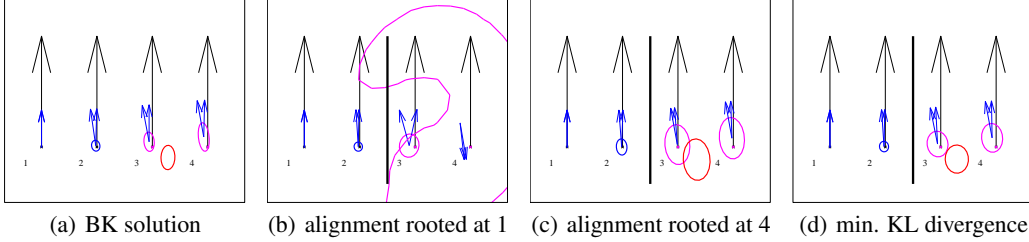

|  (a) BK solution | (b) alignment rooted at 1 | (c) alignment rooted at 4 | (d) min. KL divergence |

Figure 1: Alignment results after partition (shown by vertical line). circles represent 95% confidence intervals in the estimate of the camera location. (a) The exact solution, computed by the BK algorithm in the absence of partitions. (b) Solution obtained when aligning from node 1. (c) Solution obtained when aligning from node 4. (d) Solution obtained by joint optimized alignment.

## 5 Robust distributed filtering

In the previous section, we introduced an algorithm for distributed filtering with dynamic Bayesian networks that, with sufficient communication, converges to the centralized B&K98 algorithm. In some settings, for example when interference causes a network partition, messages may not be propagated long enough to guarantee convergence before nodes must roll-up to the next time step. Consider the example, illustrated in Figure 1, in which a network of cameras localizes itself by observing a moving object. Each camera $i$ carries a clique marginal over the location of the object $M^{(t)}$, its own camera pose variable $C_i$, and the pose of one of its neighboring cameras: $\pi_1(C_{1,2}, M^{(t)})$, $\pi_2(C_{2,3}, M^{(t)})$, and $\pi_3(C_{3,4}, M^{(t)})$. Suppose communication were interrupted due to a network partition: observations would not propagate, and the marginals carried by the nodes would no longer form a consistent distribution, in the sense that $\pi_1, \pi_2, \pi_3$ might not agree on their marginals, e.g., $\pi_1(C_2, M^{(t)}) \neq \pi_2(C_2, M^{(t)})$. The goal of **alignment** is to obtain a consistent distribution $\tilde{p}(\mathbf{X}^{(t)} \mid \mathbf{z}^{(1:t-1)})$ from marginals $\pi_1, \pi_2, \pi_3$ that is close to the true posterior $p(\mathbf{X}^{(t)} \mid \mathbf{z}^{(1:t-1)})$ (as measured, for example, by the root-mean-square error of the estimates). For simplicity of notation, we omit time indices $t$ and conditioning on the past evidence $\mathbf{z}^{(1:t-1)}$ throughout this section.

### 5.1 Optimized conditional alignment

One way to define a consistent distribution $\tilde{p}$ is to start from a root node $r$, e.g., 1, and allow each clique marginal to decide the conditional density of $\mathbf{C}_i$ given its parent, e.g.,

$$\tilde{p}_1(C_{1:4}, M) = \pi_1(C_{1,2}, M) \times \pi_2(C_3 \mid C_2, M) \times \pi_3(C_4 \mid C_3, M).$$

This density $\tilde{p}_1$ forms a coherent distribution over $C_{1:4}, M$, and we say that $\tilde{p}_1$ is **rooted** at node 1. Thus, $\pi_1$ fully defines the marginal density over $C_{1,2}, M$, $\pi_2$ defines the conditional density of $C_3$ given $C_2, M$, and so on. If node 3 were the root, then node 1 would only contribute $\pi_1(C_1 \mid C_2, M)$, and we would obtain a different approximate distribution.

In general, given a collection of marginals $\pi_i(\mathbf{C}_i)$ over the cliques of a junction tree $T$, and a root node $r \in N_T$, the distribution obtained by **conditional alignment** from $r$ can be written as

$$\tilde{p}_r(\mathbf{X}) = \pi_r(\mathbf{C}_r) \times \prod_{i \in (N_T - \{r\})} \pi_i(\mathbf{C}_i - \mathbf{S}_{up(i),i} \mid \mathbf{S}_{up(i),i}), \tag{2}$$

where $up(i)$ denotes the upstream neighbor of $i$ on the (unique) path between $r$ and $i$.

The choice of the root $r$ often crucially determines how well the aligned distribution $\tilde{p}_r$ approximates the true prior. Suppose that, in the example in Figure 1, the nodes on the left side of the partition do not observe the person while the communication is interrupted, and the prior marginals $\pi_1$, $\pi_2$ are uncertain about $M$. If we were to align the distribution from $\pi_2$, multiplying $\pi_3(C_4 \mid C_3, M)$ into the marginal $\pi_2(C_{2,3}, M)$ would result in a distribution that is uncertain in both $M$ and $C_4$ (Figure 1(b)), while a better choice of root could provide a much better estimate (Figure 1(c)).

One possible metric to optimize when choosing the root $r$ for the alignment is the entropy of the resulting distribution $\tilde{p}_r$. For example, the entropy of $\tilde{p}_2$ in the previous example can be written as

$$H_{\tilde{p}_2}(C_{1:4}, M) = H_{\pi_2}(C_{2,3}, M) + H_{\pi_3}(C_4 \mid C_3, M) + H_{\pi_1}(C_1 \mid C_2, M), \tag{3}$$

where we use the fact that, for Gaussians, the conditional entropy of $C_4$ given $C_3, M$ only depends on the conditional distribution $\tilde{p}_2(C_4 \mid C_3, M) = \pi_3(C_4 \mid C_3, M)$. A naïve algorithm for obtaining the best root would exploit this decomposition to compute the entropy of each $\tilde{p}_2$, and pick the root that leads to a lowest total entropy; the running time of this algorithm is $O(|N_T|^2)$. We propose a dynamic programming approach that significantly reduces the running time. Comparing Equation 3

with the entropy of the distribution rooted at a neighboring node 3, we see that they share a common term $H_{\pi_1}(C_1 \mid C_2, M)$, and $H_{\tilde{p}_3}(C_{1:4}, M) - H_{\tilde{p}_2}(C_{1:4}, M) = H_{\pi_3}(\mathbf{S}_{2,3}) - H_{\pi_2}(\mathbf{S}_{2,3}) \triangleq \triangle_{2,3}$. If $\triangle_{2,3}$ is positive, node 2 is a better root than 3, $\triangle_{2,3}$ is negative, we have the reverse situation. Thus, when comparing neighboring nodes as root candidates, the difference in entropy of the resulting distribution is simply the difference in entropy their local distributions assign to their separator. This property generalizes to the following dynamic programming algorithm that determines the root $r$ with minimal $H_{\tilde{p}_r}(\mathbf{X})$ in $O(|N_T|)$ time:

- For any node $i \in N_T$, define the message from $i$ to its neighbor $j$ as

$$m_{i \to j} = \begin{cases} \triangle_{i,j} & \text{if } m_{k \to i} < 0, \ \ \forall k \neq j \\ \triangle_{i,j} + \max_{k \neq j} m_{k \to i} & \text{otherwise} \end{cases},$$

where $\triangle_{i,j} = H_{\pi_j}(\mathbf{S}_{i,j}) - H_{\pi_i}(\mathbf{S}_{i,j})$, and $k$ varies over the neighbors of $i$ in $T$.

- If $\max_k m_{k \to i} < 0$ then $i$ is the optimal root; otherwise, $up(i) = \operatorname{argmax}_k m_{k \to i}$.

Intuitively, the message $m_{i \to j}$ represents the loss (entropy) with root node $j$, compared to the best root on $i$'s side of the tree. Ties between nodes, if any, can be resolved using node IDs.

## 5.2 Distributed optimized conditional alignment

In the absence of an additional procedure, RDPI can be viewed as performing conditional alignment. However, the alignment is applied to the local belief at each node, rather than the global distribution, and the nodes may not agree on the choice of the root $r$. Thus, the network is not guaranteed to reach a globally consistent, aligned distribution. In this section, we show that RDPI can be extended to incorporate the optimized conditional alignment (OCA) algorithm from the previous section.

By Property 1, at convergence, the priors at each node form a subtree of an external junction tree for the assumed density. Conceptually, if we were to apply OCA to this subtree, the node would have an aligned distribution, but nodes may not be consistent with each other. Intuitively, this happens because the optimization messages $m_{i \to j}$ were not propagated between different nodes.

In RDPI, node $n$'s belief $\beta_n$ includes a collection of (potentially inconsistent) priors $\{\pi_i(\mathbf{C}_i)\}$. In the standard sum-product inference algorithm, an inference message $\mu_{m \to n}$ from node $m$ to node $n$ is computed by marginalizing out some variables from the factor $\mu_{m \to n}^+ \triangleq \psi_m \times \prod_{k \neq n} \mu_{k \to m}$ that combines the messages received from node $m$'s other neighbors with node $m$'s local belief. The inference message in RDPI involves a similar marginalization, which corresponds to **pruning** some cliques from $\mu_{m \to n}^+$ [7]. When such pruning occurs, any likelihood information $\lambda_i(\mathbf{C}_i)$ associated with the pruned clique $i$ is transferred to its neighbor $j$.

Our distributed OCA algorithm piggy-backs on this pruning, computing an optimization message $m_{i \to j}$, which is stored in clique $j$. (To compute this message, cliques must also carry their original, unaligned priors.) At convergence, the nodes will not only have a subtree of an external tree, but also the incoming optimization messages that result from pruning of all other cliques of the external tree. In order to determine the globally optimal root, each node (locally) selects a root for its subtree. If this root is one of the initial cliques associated with $n$, then $n$, and in particular this clique, is the root of the conditional alignment. The alignment is propagated throughout the network. If the optimal root is determined to be a clique that came from a message received from a neighbor, then the neighbor (or another node upstream) is the root, and node $n$ aligns itself with respect to the neighbor's message. With an additional tie-breaking rule that ensures that all the nodes make consistent choices about their subtrees [4], this procedure is equivalent to running the OCA algorithm centrally:

**Theorem 2.** *Given sufficient communication and in the absence of network partitions, nodes running distributed* OCA *reach a globally consistent belief based on conditional alignment, selecting the root clique that leads to the joint distribution of minimal entropy. In the presence of partitions, each partition will reach a consistent belief that minimizes the entropy within this partition.*

## 5.3 Jointly optimized alignment

While conceptually simple, there are situations where such a rooted alignment will not provide a good aligned distribution. For example, if in the example in Figure 1, cameras 2 and 3 carry marginals $\pi_2(C_{2,3}, M)$ and $\pi_{2'}(C_{2,3}, M)$, respectively, and both observe the person, node 2 will have a better estimate of $C_2$, while node 3's estimate of $C_3$ will be more accurate. If either node is chosen as the root, the aligned distribution will have a worse estimate of the pose of one of the cameras, because performing rooted alignment from either direction effectively overwrites the marginal of the other node. In this example, rather than fixing a root, we want an aligned distribution that attempts to simultaneously optimize the distance to both $\pi_2(C_{2,3}, M)$ and $\pi_{2'}(C_{2,3}, M)$.

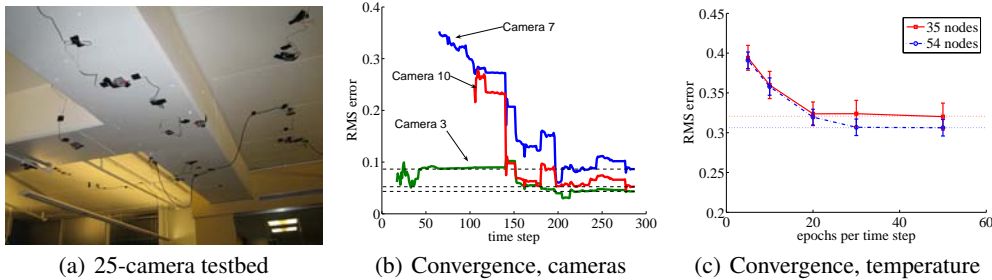

| (a) 25-camera testbed | (b) Convergence, cameras | (c) Convergence, temperature |

Figure 2: (a) Testbed of 25 cameras used for the SLAT experiments. (b) Convergence results for individual cameras in one experiment. Horizontal lines indicate the cooresponding centralized solution at the end of the experiment. (c) Convergence versus amount of communication for a temperature network of 54 real sensors.

We propose the following optimization problem that minimizes the sum of reverse KL divergence from the aligned distribution to the clique marginals $\pi_i(\mathbf{C}_i)$:

$$\tilde{p}(\mathbf{X}) = \operatorname*{argmin}_{q(\mathbf{X}), q \models T} \sum\nolimits_{i \in N_T} D(q(\mathbf{C}_i) \,\|\, \pi_i(\mathbf{C}_i)),$$

where $q \models T$ denotes the constraint that $\tilde{p}$ factorizes according to the junction tree $T$. This method will often provide very good aligned distributions (e.g., Figure (d)). For Gaussian distributions, this optimization problem corresponds to

$$\min_{\mu_{\mathbf{C}_i}, \Sigma_{\mathbf{C}_i}} \quad \sum\nolimits_{i \in N_T} -\log|\Sigma_{\mathbf{C}_i}| + \langle \Sigma_i^{-1}, \Sigma_{\mathbf{C}_i} \rangle + \sum\nolimits_{i \in N_T} (\mu_i - \mu_{\mathbf{C}_i})^T \Sigma_i^{-1} (\mu_i - \mu_{\mathbf{C}_i}),$$

$$\text{subject to} \quad \Sigma_{\mathbf{C}_i} \succeq 0, \quad \forall i \in N_T, \tag{4}$$

where $\mu_{\mathbf{C}_i}, \Sigma_{\mathbf{C}_i}$ are the means and covariances of $q$ over the variables $\mathbf{C}_i$, and $\mu_i, \Sigma_i$ are the means and covariances of the marginals $\pi_i$. The problem in Equation 4 consists of two independent convex optimization problems over the means and covariances of $q$, respectively. The former problem can be solved in a distributed manner using distributed linear regression [6], while the latter can be solved using a distributed version of an iterative methods, such as conjugate gradient descent [1].

## 6    Experimental results

We evaluated our approach on two applications: a camera localization problem [5] (SLAT), in which a set of cameras simultaneously localizes itself by tracking a moving object, and temperature monitoring application, analogous to the one presented in [7]. Figure 2(a) shows some of the 25 ceiling-mounted cameras used to collect the data in our camera experiments. We implemented our distributed algorithm in a network simulator that incorporates message loss and used data from these real sensors as our observations. Figure 2(b) shows the estimates obtained by three cameras in one of our experiments. Note that each camera converges to the estimate obtained by the centralized B&K98 algorithm. In Figure 2(c), we evaluate the sensitivity of the algorithm to incomplete communication. We see that, with a modest number of rounds of communication performed in each time step, the algorithm obtains a high quality of the solution and converges to the centralized solution.

In the second set of experiments, we evaluate the alignment methods, presented in Section 5. In Figure 3(a), the network is split into four components; in each component, the nodes communicate fully, and we evaluate the solution if the communication were to be restored after a given number of time steps. The vertical axis shows the RMS error of estimated camera locations at the end of the experiment. For the unaligned solution, the nodes may not agree on the estimated pose of a camera, so it is not clear which node's estimate should be used in the RMS computation; the plot shows an "omniscient envelope" of the RMS error, where, given the (unknown) true camera locations, we select the best and worst estimates available in the network for each camera's pose. The results show that, in the absence of optimized alignment, inconsistencies can degrade the solution: observations collected after the communication is restored may not make up for the errors introduced by the partition.

The third experiment evaluates the performance of the distributed algorithm in highly-disconnected scenarios. Here, the sensor network is hierarchically partitioned into smaller disconnected components by selecting a random cut through the largest component. The communication is restored shortly before the end of the experiment. Figures 3(b) shows the importance of aligning from the correct node: the difference between the **optimized root** and an arbitrarily chosen root is significant, particularly when the network becomes more and more fractured. In our experiments, large errors often resulted from the nodes having uncertain beliefs, hence justifying the objective function. We see that the jointly optimized alignment described in Section 5.3, **min. KL**, tends to provide the best aligned distribution, though often close to the optimized root, which is simpler

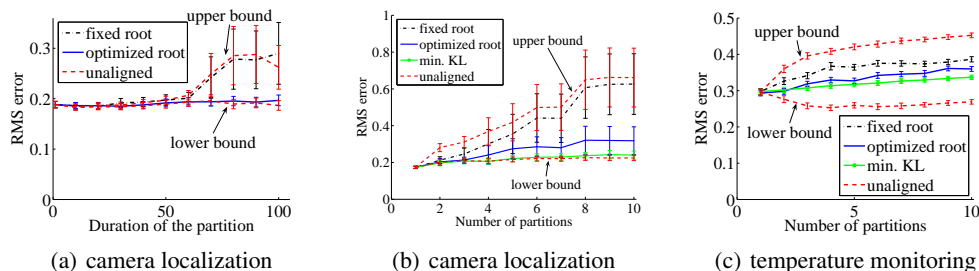

(a) camera localization      (b) camera localization      (c) temperature monitoring

Figure 3: Comparison of the alignment methods. (a) RMS error vs. duration of the partition. For the unaligned solution, the plot shows bounds on the error: given the (unknown) camera locations, we select the best and worst estimates available in the network for each camera's pose. In the absence of optimized alignment, inconsistencies can degrade the quality of the solution. (b, c) RMS error vs. number of partitions. In camera localization (b), the difference between the optimized alignment and the alignment from an arbitrarily chosen fixed root is significant. For the temperature monitoring (c), the differences are less pronounced, but follow the same trend.

to compute. Finally, 3(c) shows the alignment results on the temperature monitoring application. Compared to SLAT, the effects of network partitions on the results for the temperature data are less severe. One contributing factor is that every node in a partition is making local temperature observations, and the approximate transition model for temperatures in each partition is quite accurate, hence all the nodes continue to adjust their estimates meaningfully while the partition is in progress.

## 7  Conclusions

This paper presents a new distributed approach to approximate dynamic filtering based on a distributed representation of the assumed density in the network. Distributed filtering is performed by first conditioning on evidence using a robust distributed inference algorithm [7], and then advancing to the next time step locally. With sufficient communication in each time step, our distributed algorithm converges to the centralized B&K98 solution. In addition, we identify a significant challenge for probabilistic inference in dynamical systems: nodes can have inconsistent beliefs about the current state of the system, and an ineffective handling of this situation can lead to very poor estimates of the global state. We address this problem by developing a distributed algorithm that obtains an informative consistent distribution, optimizing over various choices of the root node, and an alternative joint optimization approach that minimizes a KL divergence-based criterion. We demonstrate the effectiveness of our approach on a suite of experimental results on real-world sensor data.

### Acknowledgments

This research was supported by grants NSF-NeTS CNS-0625518 and CNS-0428738 NSF ITR. S. Funiak was supported by the Intel Research Scholar Program; C. Guestrin was partially supported by an Alfred P. Sloan Fellowship.

## References

[1] D. P. Bertsekas and J. N. Tsitsiklis. *Parallel and Distributed Computation: Numerical Methods*. Athena Scientific; 1st edition (January 1997), 1997.

[2] X. Boyen and D. Koller. Tractable inference for complex stochastic processes. In *Proc. of UAI*, 1998.

[3] R. Cowell, P. Dawid, S. Lauritzen, and D. Spiegelhalter. *Probabilistic Networks and Expert Systems*. Springer, New York, NY, 1999.

[4] S. Funiak, C. Guestrin, M. Paskin, and R. Sukthankar. Robust probabilistic filtering in distributed systems. Technical Report CMU-CALD-05-111, Carnegie Mellon University, 2005.

[5] S. Funiak, C. Guestrin, M. Paskin, and R. Sukthankar. Distributed localization of networked cameras. In *Proc. of Fifth International Conference on Information Processing in Sensor Networks (IPSN-06)*, 2006.

[6] C. Guestrin, R. Thibaux, P. Bodik, M. A. Paskin, and S. Madden. Distributed regression: an efficient framework for modeling sensor network data. In *Proc. of IPSN*, 2004.

[7] M. A. Paskin and C. E. Guestrin. Robust probabilistic inference in distributed systems. In *UAI*, 2004.

[8] M. A. Paskin, C. E. Guestrin, and J. McFadden. A robust architecture for inference in sensor networks. In *Proc. of IPSN*, 2005.

[9] A. Pfeffer and T. Tai. Asynchronous dynamic Bayesian networks. In *Proc. UAI 2005*, 2005.

[10] M. Rosencrantz, G. Gordon, and S. Thrun. Decentralized sensor fusion with distributed particle filters. In *Proc. of UAI*, 2003.

[11] F. Zhao, J. Liu, J. Liu, L. Guibas, and J. Reich. Collaborative signal and information processing: An information directed approach. *Proceedings of the IEEE*, 91(8):1199–1209, 2003.
